# Estimating time-varying input signals and ion channel states from a single voltage trace of a neuron

**Ryota Kobayashi**\*
Department of Human and Computer Intelligence, Ritsumeikan University
Siga 525-8577, Japan
kobayashi@cns.ci.ritsumei.ac.jp

**Yasuhiro Tsubo**
Laboratory for Neural Circuit Theory, Brain Science Institute, RIKEN
2-1 Hirosawa Wako, Saitama 351-0198, Japan
yasuhirotsubo@riken.jp

**Petr Lansky**
Institute of Physiology, Academy of Sciences of the Czech Republic
Videnska 1083, 142 20 Prague 4, Czech Republic
lansky@biomed.cas.cz

**Shigeru Shinomoto**
Department of Physics, Kyoto University
Kyoto 606-8502, Japan
shinomoto@scphys.kyoto-u.ac.jp

## Abstract

State-of-the-art statistical methods in neuroscience have enabled us to fit mathematical models to experimental data and subsequently to infer the dynamics of hidden parameters underlying the observable phenomena. Here, we develop a Bayesian method for inferring the time-varying mean and variance of the synaptic input, along with the dynamics of each ion channel from a single voltage trace of a neuron. An estimation problem may be formulated on the basis of the state-space model with prior distributions that penalize large fluctuations in these parameters. After optimizing the hyperparameters by maximizing the marginal likelihood, the state-space model provides the time-varying parameters of the input signals and the ion channel states. The proposed method is tested not only on the simulated data from the Hodgkin−Huxley type models but also on experimental data obtained from a cortical slice *in vitro*.

## 1 Introduction

Owing to the great advancements in measurement technology, a huge amount of data is generated in the field of science, engineering, and medicine, and accordingly, there is an increasing demand for estimating the hidden states underlying the observed signals. Neurons transmit information by transforming synaptic inputs into action potentials; therefore, it is essential to investigate the dynamics of the synaptic inputs to understand the mechanism of the information processing in neuronal systems. Here we propose a method to deduce the dynamics from experimental data.

Cortical neurons *in vivo* receive synaptic bombardments from thousands of neurons, which cause the membrane voltage to fluctuate irregularly. As each synaptic input is small and the synaptic input rate is high, the total input can be characterized only with its mean and variance, as in the mathematical description of Brownian motion of a small particle suspended in a fluid. Given the information of the mean and variance of the synaptic input, it is possible to estimate the underlying excitatory and inhibitory firing rates from respective populations of neurons.

The membrane voltage fluctuations in a neuron are caused not only by the synaptic input but also by the hidden dynamics of ionic channels. These dynamics can be described by conductance-based models, including the Hodgkin−Huxley model. Many studies have been reported on the dynamics of ionic channels and their impact on neural coding properties [1].

There have been attempts to decode a voltage trace in terms of input parameters; the maximum likelihood estimator for current inputs was derived under an assumption of linear leaky integration [2, 3]. Empirical attempts were made to infer conductance inputs by fitting an approximate distribution of the membrane voltage to the experimental data [4, 5]. A linear regression method was proposed to infer the maximal ionic conductances and single synaptic inputs in the dendrites [6]. In all studies, these input parameters were assumed to be constant in time. In practice, however, such assumption of the constancy of input parameters is too strong simplification for the neuronal firing [7, 8].

In this paper, we propose a method for the simultaneous identification of the time-varying input parameters and of the ion-channels dynamics from a single voltage trajectory. The problem is ill-posed, in the sense that the set of parameters giving rise to a particular voltage trace cannot be uniquely determined. However, the problem may be formulated as a statistical problem of estimating the hidden state using a state-space model and then it is solvable. We verify the proposed method by applying it not only to numerical data obtained from the Hodgkin−Huxley type models but also to the biological data obtained in *in vitro* experiment.

## 2 Model

### 2.1 Conductance-based model

We start from the conductance-based neuron model [1]:

$$\frac{dV}{dt} = -\bar{g}_{\text{leak}}(V - E_{\text{leak}}) - \sum_{\text{ion}} J_{\text{ion}}(V, \vec{w}) + J_{\text{syn}}(t), \tag{1}$$

where, $\bar{g}_{\text{leak}} =: g_{\text{leak}}/C_m$, $J_{\text{ion}} =: I_{\text{ion}}/C_m$, $J_{\text{syn}}(t) := I_{\text{syn}}(t)/C_m$,

$V$ is the membrane voltage, $\bar{g}_{\text{leak}}$ is the normalized leak conductance, $E_{\text{leak}}$ is the reversal potential, $J_{\text{ion}}$ are the voltage-dependent ionic inputs, $\vec{w} := (w_1, w_2, \cdots, w_d)$ are the gating variables that characterize the states of ion channels, $J_{\text{syn}}$ is a synaptic input, $C_m$ is the membrane capacitance, $I_{\text{ion}}$ are the voltage-dependent ionic currents and $I_{\text{syn}}(t)$ is a synaptic input current. The ionic inputs $J_{\text{ion}}$ are a nonlinear function of $V$ and $\vec{w}$. Each gating variable $w_i$ ($i = 1, \cdots, d$) follows the Langevin equation [9]:

$$\frac{dw_i}{dt} = \alpha_i(V)(1 - w_i) - \beta_i(V)w_i + s_i\xi_i(t), \tag{2}$$

where $\alpha_i(V)$, $\beta_i(V)$ are nonlinear functions of the voltage, $s_i$ is the standard deviation of the channel noise, and $\xi_i(t)$ is an independent Gaussian white noise with zero mean and unit variance. The synaptic input $J_{\text{syn}}(t)$ is the sum of the synaptic inputs from a large number of presynaptic neurons. If each synaptic input is weak and the synaptic time constants are small, we can adopt a diffusion approximation [10],

$$J_{\text{syn}}(t) = \mu(t) + \sigma(t)\chi(t), \tag{3}$$

where $\mu(t)$, $\sigma(t)$ are the instantaneous mean and standard deviation of the synaptic input, and $\chi(t)$ is Gaussian white noise with zero mean and unit variance. The components $\mu(t)$ and $\sigma^2(t)$ are considered to be the input signals to a neuron.

### 2.2 Estimation Problem

The problem is to find the parameters of model (1-3) from a single voltage trace $\{V(t)\}$. There are three kinds of parameters in the model. The first kind is the input signals $\{\mu(t), \sigma^2(t)\}$. The second

kind is the gating variables $\{\vec{w}(t)\}$ that characterize the activity of the ionic channels. The remaining parameters are the intrinsic parameters of a neuron, such as the standard deviation of the channel noise, the functional form of voltage-dependent ionic inputs, and that of the rate constants. Some of these parameters, i.e., $J_{\text{ion}}(V, \vec{w})$, $\alpha_i(V)$, $\beta_i(V)$, $\bar{g}_{\text{leak}}$ and $E_{\text{leak}}$ are measurable by additional experiments. After determining such intrinsic parameters of the third group by separate experiments, we estimate parameters of the first and second group from a single voltage trace.

## 3 Method

Because of the ill-posedness of the estimation problem, we cannot determine the input signals from a voltage trace alone. To overcome this, we introduce random-walk-type priors for the input signals. Then, we determine hyperparameters using the EM algorithm. Finally, we evaluate the Bayesian estimate for the input signals and the ion channel states with the Kalman filter and smoothing algorithm. Figure 1 is a schematic of the estimation method.

### 3.1 Priors for Estimating Input Parameters

Let us assume, for the sake of simplicity, that the voltage is sampled at $N$ equidistant steps $\delta t$, denoting by $V_j$ the observed voltage at time $j\delta t$. To apply the Bayesian approach, the conductance based model (1, 3) is modified into the discretized form:

$$V_{j+1} = V_j + \left\{ -\bar{g}_{\text{leak}}(V_j - E_{\text{leak}}) - \sum_{\text{ion}} J_{\text{ion}}(V_j, \vec{w}_j) + M_j \right\} \delta t + \sqrt{S_j \delta t}\, \eta_j, \qquad (4)$$

where $\{M_j, S_j\}$ are random functions of time, $\eta_j$ is a standard Gaussian random variable. It is not possible to infer a large set of parameters $\{M_j, S_j\}$ from a single voltage trace $\{V_j\}$ alone, because the number of parameters overwhelms the number of data points. To resolve it, we introduce random-walk-type priors, i.e. we assume that the random functions are sufficiently smooth to satisfy the following conditions [11]:

$$P[M_{j+1}|M_j = m] \sim N(m, \gamma_M^2 \delta t), \qquad (5)$$

$$P[S_{j+1}|S_j = s] \sim N(s, \gamma_S^2 \delta t), \qquad (6)$$

where $\gamma_M$ and $\gamma_S$ are hyperparameters that regulate the smoothness of $M(t)$ and $S(t)$, respectively, and $N(\mu, \sigma^2)$ represents the Gaussian distribution with mean $\mu$ and variance $\sigma^2$.

### 3.2 Formulation as a State Space model

The model described in the previous sections could be represented as the state-space model, in which $\vec{x}_j \equiv (M_j, S_j, \vec{w}_j)$ are the $(d+2)$-dimensional states, and $Z_j \equiv V_{j+1} - V_j$ $(j = 1, \cdots, N-1)$ are the observations. The kinetic equations (2) and the prior distributions (5, 6) can be rewritten as

$$\vec{x}_{j+1} = F_j \vec{x}_j + \vec{u}_j + G\vec{\eta}_j, \qquad (7)$$

where

$$F_j = \text{diag}(1, 1, a_{1;j}, a_{2;j}, \cdots, a_{d;j}), \; G = \text{diag}(\gamma_M \sqrt{\delta t}, \gamma_S \sqrt{\delta t}, s_1 \sqrt{\delta t}, s_2 \sqrt{\delta t}, \cdots, s_d \sqrt{\delta t}),$$
$$\vec{u}_j = (0, 0, b_{1;j}, b_{2;j}, \cdots, b_{d;j})^T,$$

$F_j$ and $G$ are $(d+2) \times (d+2)$ diagonal matrices, $\vec{u}_j$ is $(d+2)$-dimensional vector, and $\vec{\eta}_j$ is a $(d+2)$-dimensional independent Gaussian random vector with zero mean and unit variance. $a_{i,j}$ and $b_{i,j}$ is given by

$$a_{i,j} = 1 - \{\alpha_i(V_j) + \beta_i(V_j)\}\delta t, \; b_{i,j} = \alpha_i(V_j)\delta t,$$

The observation equation is obtained from Eq. (4):

$$Z_j = -\bar{g}_{\text{leak}}(V_j - E_{\text{leak}})\delta t - \sum_{\text{ion}} J_{\text{ion}}(V_j, \vec{w}_j)\delta t + M_j \delta t + \sqrt{S_j \delta t}\, \xi_j, \qquad (8)$$

where $\xi_j$ is an independent Gaussian random variable with zero mean and unit variance. In the estimation problem, only $\{V_j\}_{j=1}^N$ are observable. $\{\vec{x}_j\}_{j=1}^N$ are the hidden variables because it cannot be observed in a experiment.

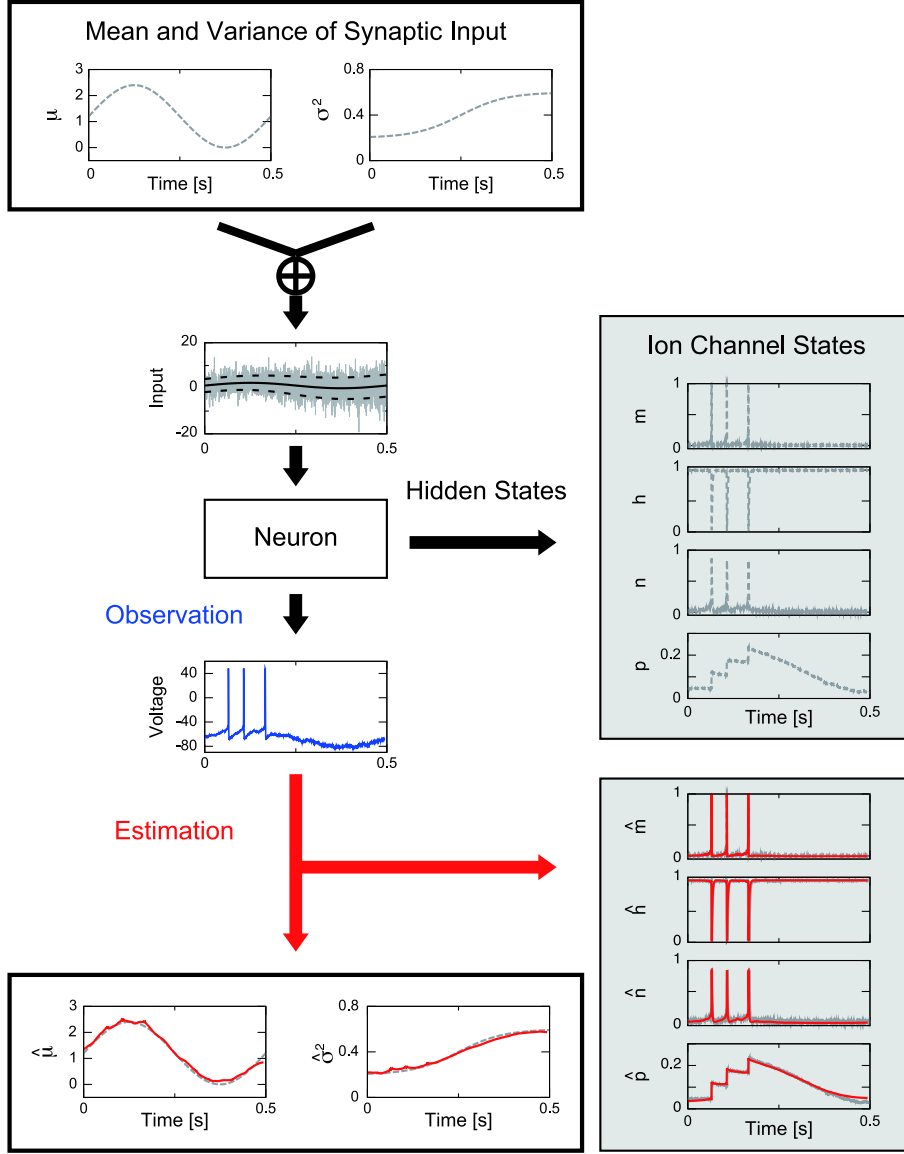

Figure 1: A schema of the estimation procedure: A conductance-based model neuron [12] is driven by a fluctuating input of the mean $\mu(t)$ and variance $\sigma^2(t)$ varying in time. The $\mu(t)$ (black line) and the $\mu(t) \pm \sigma(t)$ (black dotted lines) are depicted in the second panel from the top. We estimate the input signals $\{\mu(t), \sigma^2(t)\}$ and the gating variables $\{m(t), h(t), n(t), p(t)\}$ from a single voltage trace (blue line). The estimated results are shown in the bottom panels. The input signals are in the two panels and the ion channel states are in the right shaded box. Gray dashed lines are the true values and red lines are their estimates.

### 3.3 Hyperparameter Optimization

We determine $d + 2$ hyperparameters $\vec{q} := (\gamma_M^2, \gamma_S^2, s_1^2, \cdots, s_d^2)$ by maximizing the marginal likelihood via the EM algorithm [13]. We maximize the likelihood integrated over hidden variables $\{\vec{x}_t\}_{t=1}^{N-1}$,

$$\vec{q}_{\mathrm{ML}} = \underset{\vec{q}}{\operatorname{argmax}} \, p(Z_{1:N-1}|\vec{q}) = \underset{\vec{q}}{\operatorname{argmax}} \int p(Z_{1:N-1}, \vec{x}_{1:N-1}|\vec{q}) d\vec{x}_{1:N-1}, \qquad (9)$$

where $Z_{1:N-1} := \{Z_j\}_{j=1}^{N-1}$, $\vec{x}_{1:N-1} := \{\vec{x}_j\}_{j=1}^{N-1}$, and $d\vec{x}_{1:N-1} := \Pi_{j=1}^{N-1} d\vec{x}_j$. The maximization can be achieved by iteratively maximizing the $Q$ function, the conditional expectation of the log likelihood:

$$\vec{q}_{k+1} = \underset{\vec{q}}{\operatorname{argmax}}\, Q(\vec{q}|\vec{q}_k), \tag{10}$$

$$\text{where } Q(\vec{q}|\vec{q}_k) := E[\log(P[Z_{1:N-1}, \vec{x}_{1:N-1}|\vec{q}])|Z_{1:N-1}, \vec{q}_k],$$

$\vec{q}_k$ is the $k$th iterated estimate of $\vec{q}$, $E[X|Y]$ is the conditional expectation of $X$ given the value of $Y$, and $P[X|Y]$ is the conditional probability distribution of $X$ given the value of $Y$.
The $Q$ function can be written as

$$Q(\vec{q}|\vec{q}_k) = \sum_{j=1}^{N-1} E[\log(P[Z_j|\vec{x}_j]) \,|Z_{1:N-1}, \vec{q}_k] + \sum_{j=1}^{N-2} E[\log(P[\vec{x}_{j+1}|\vec{x}_j, \vec{q}]) \,|Z_{1:N-1}, \vec{q}_k]. \tag{11}$$

The $(k+1)$ th iterated estimate of $\vec{q}$ is determined by the conditions for $\partial Q/\partial q_i = 0$:

$$q_{i,k+1} = \frac{1}{(N-2)\delta t} \sum_{j=1}^{N-2} E[(x_{i,j+1} - f_{i,j}x_{i,j} - u_{i,j})^2 | Z_{1:N-1}, \vec{q}_k], \tag{12}$$

where $q_{i,k+1}$ is the $i$th component of the $\vec{q}_{k+1}$, $x_{i,j}$ is the $i$th component of $\vec{x}_j$, $f_{i,j}$ is the $i$th diagonal component of $F_j$, and $u_{i,j}$ is the $i$th component of $\vec{u}_j$. As the EM algorithm increases the marginal likelihood at each iteration, the estimate converges to a local maximum. We calculate the conditional expectations in Eq.(12) using Kalman filter and smoothing algorithm [11, 14, 15, 16, 17].

### 3.4 Bayesian estimator for the input signal

After fitting the hyperparameters, we evaluate the Bayesian estimator for the input signals and the gating variables:

$$\vec{x}_j^* = E[\vec{x}_j|Z_{1:N-1}, \vec{q}], \tag{13}$$

where $\vec{x}_j^*$ is the Bayesian estimator for $\vec{x}_j$. Using this estimator, we can estimate not only the (smoothly) time-varying mean and variance of the synaptic input $\{\mu(t), \sigma^2(t)\}$, but also the time evolution of the gating variables $\vec{w}(t)$. We evaluate the estimator (13) using a Kalman filter and smoothing algorithm [11, 14, 15, 16, 17].

## 4 Applications

### 4.1 Estimating time-varying input signals and ion channel states in a conductance-based model

To test the accuracy and robustness of our method, we applied the proposed method to simulated voltage traces. We adopted a Hodgkin−Huxley model with microscopic description of ionic channels [18], which consists of two ionic inputs $J_{\text{ion}}$ (ion $\in \{\text{Na}, \text{Kd}\}$): $J_{\text{Na}} = \gamma_{\text{Na}}[\mathbf{m}_3\mathbf{h}_1](V - E_{\text{Na}})$ and $J_{\text{Kd}} = \gamma_{\text{K}}[\mathbf{n}_4](V - E_{\text{K}})$, where $\gamma_{\text{Na(K)}}$ is the conductance of a single sodium (potassium) ion channel in the open state, $[\mathbf{m}_3\mathbf{h}_1]$ ($[\mathbf{n}_4]$) is the number of sodium (potassium) channels that are open and $E_{\text{Na(K)}}$ is the sodium (potassium) reversal potential. There are 8 (5) states in a sodium (potassium) channel and the state transitions are described by a Markov chain model. Details of this model can be found in [18].

First, we apply the proposed method to sinusoidally modulated input signals. Figure 2B compares the time-varying input signals $\{\mu(t), \sigma^2(t)\}$ with their estimate and Figure 2C compares the open probability of each ion channel with its estimate. It is observed in this case that the method provides the accurate estimate. Second, we examine whether the method can also work in the presence of discontinuity in the input signals. Though discontinuous inputs do not satisfy the smoothness assumption (5, 6), the method gives accurate estimates (Figure 3A). Third, the estimation method is applied to conductance input model, which is given by $J_{\text{syn}}(t) = \bar{g}_E \sum_{j,k} \delta(t - t_{E,j}^k)(V_E - V(t)) + \bar{g}_I \sum_{j,k} \delta(t - t_{I,j}^k)(V_I - V(t))$, where the subscript $E(I)$ means the excitatory (inhibitory) synapse,

$\bar{g}_{E(I)}$ is the normalized postsynaptic conductance, $V_{E(I)}$ is the reversal potential and $t^k_{E(I),j}$ is the $k$th spike time of the $j$th presynaptic neuron, and $\delta(t)$ is the Dirac delta function. It can be seen from Figure 3B that the method can provide accurate estimate except during action potentials when the input undergoes a rapid modulation. Fourth, the effect of observation noise on the estimation accuracy is investigated. We introduce an observation noise in the following manner: $Z_{\mathrm{obs},j} = Z_j + \sigma_{\mathrm{obs}}\eta_j$, where $Z_{\mathrm{obs},j} =: V_{\mathrm{obs},j+1} - V_{\mathrm{obs},j}$ is the observed value, $V_{\mathrm{obs},j}$ is the recorded voltage at time step $j$, $\sigma_{\mathrm{obs}}$ is the standard deviation of the observation noise and $\eta_j$ is an independent Gaussian random variable with zero mean and unit variance. Mathematically, it is equivalent to assume the observation noise as an additive Gaussian white noise on the voltage. In such a case, the estimation method reckons the input variance at the sum of the original input variance $\sigma^2(t)$ and the observation noise variance $\sigma^2_{\mathrm{obs}}$ (Figure 3C).

Furthermore, we also tested the present framework in its potential applicability to more complicated conductance-based models, which have slow ionic currents. To observe this, we adopted a conductance-based model proposed by Pospischil et al. [12] that has three ionic inputs $J_{\mathrm{ion}}$ (ion $\in$ $\{\mathrm{Na}, \mathrm{Kd}, \mathrm{M}\}$): $J_{\mathrm{Na}} = \bar{g}_{\mathrm{Na}} m^3 h (V - E_{\mathrm{Na}})$, $J_{\mathrm{Kd}} = \bar{g}_{\mathrm{Kd}} n^4 (V - E_{\mathrm{K}})$ and $J_{\mathrm{M}} = \bar{g}_{\mathrm{M}} p (V - E_{\mathrm{K}})$, where $\{m, h, n, p\}$ are the gating variables, $\bar{g}_{\mathrm{ion}}$ represents the normalized ionic conductances and $E_{\mathrm{ion}}$ are the reversal potentials. (See [12] for details.) An example of the estimation result is shown in Figure 1.

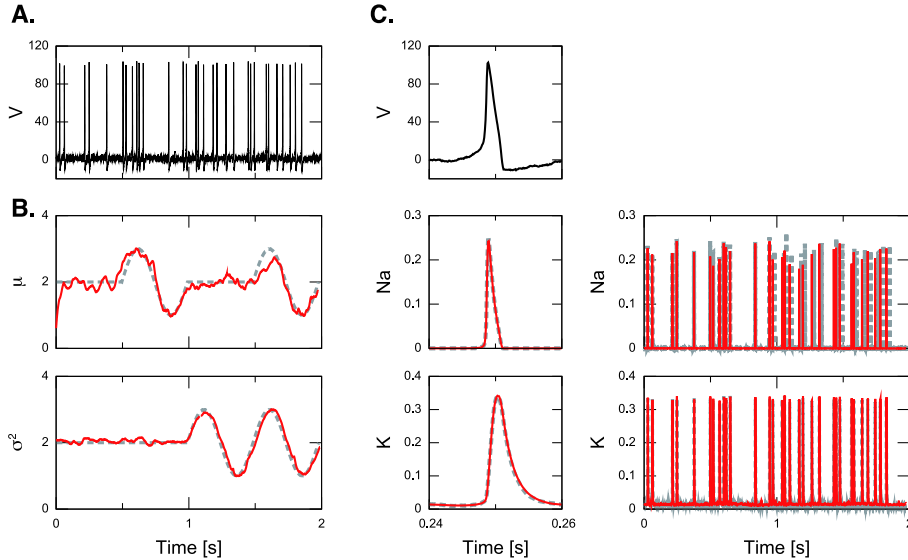

Figure 2: Estimation of input signals and ion channel states from the simulated data: A. Voltage Trace. B. Estimate of the mean $\mu$ and variance $\sigma^2$ input signals. C. Estimate of the ion channel states. The time evolution of the open probabilities of sodium (Na) and potassium (K) channels are shown. The gray dashed lines and red lines represent the true and the estimates, respectively.

## 4.2 Estimating time-varying input signals and ion channel states in experimental data

We applied the proposed method to experimental data. Randomly fluctuating current, generated by the sum of the filtered time-dependent Poisson process, was injected to a neuron in the rat motor cortex and the membrane voltage was recorded intracellularly *in vitro*. Details of the experimental procedure can be found in [19, 20]. We adopted the neuron model proposed by Pospischil et al. [12] for the membrane voltage. After tuning the ionic conductances and kinetic parameters, six hyperparameters $\gamma_{M,S}$ and $s_{m,h,n,p}$ were optimized using Eq. (12). For avoiding over-fitting, we set the upper limit $s_{\max} = 0.002$ for the hyperparameters of the gating variables. The observation noise variance was estimated from data recorded in absence of stimulation: $\sigma^2_{\mathrm{obs}} = 0.66\ [(\mathrm{mV})^2/\mathrm{ms}]$. The variance of the input signal was estimated by subtracting the observation noise variance from the estimated variance. In this way, the mean and standard deviation (SD) of the input as well as

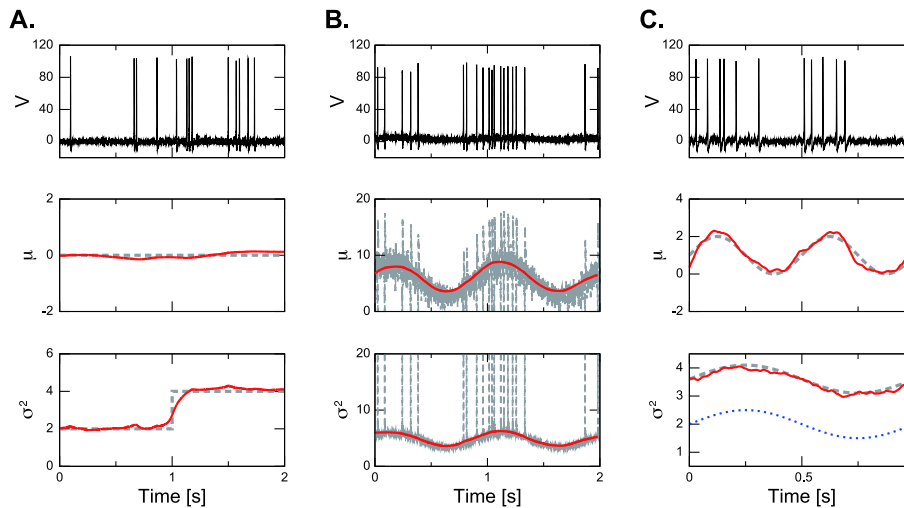

Figure 3: Robustness of the estimation method: A. Constant input with a jump. B. Conductance input. C. Sinusoidal input with observation noise. Voltage traces used for the estimation and estimates of the input signals $\{\mu(t), \sigma^2(t)\}$ are shown. In A and B, the gray dashed and red lines represents the true and the estimated input signals, respectively. In C, the blue dotted line represents the true input variance $\sigma^2(t)$, the gray dotted line represents the sum of the true input variance and the true observation noise variance $\sigma^2_{\mathrm{obs}} = 1.6$ [(mV)$^2$/ms], and the red line represents the estimated variance.

the gating variables were estimated. The time-varying mean and SD of the input are compared with their estimates in Figure 4B. The results suggest that the proposed method is applicable for these experimental data.

## 5  Discussion

We have developed a method for estimating not only the time-varying mean and variance of the synaptic input but also the ion channel states from a single voltage trace of a neuron. It was confirmed that the proposed method is capable of providing accurate estimate by applying it to simulated data. We also tested the general applicability of this method by applying it to experimental data obtained with current injection to a neuron in cortical slice preparation.

Until now, several attempts have been made to estimate synaptic input from experimental data [2, 4, 5, 8, 21, 22]. The new aspects introduced in this paper are the implementation of the state space model that allows to estimate the input signals to fluctuate in time and the gating variables that varies according to the voltage. However, the present method can be implemented under several simplifying assumptions, whose validity should be verified.

First, we approximated the synaptic inputs by white (uncorrelated) noise. In practice, the synaptic inputs are conductance-based and inevitably have the correlation of a few milliseconds. We have confirmed the applicability of the model to the numerical data generated with conductance input, and also the experimental data in which temporally correlated current is injected to a neuron. These results indicate that the white noise assumption in our method robustly applies to the reality.

Second, we constructed the state space method by assuming the smooth fluctuation of the input signals, or equivalently, by penalizing the rapid fluctuation in the prior distribution. By applying the present method to the case of stepwise change in the input signals, we realized that the method is rather robust against an abrupt change.

Third, we also approximated the channel noise by the white noise. We tested our method by applying it to a more realistic Hodgkin−Huxley type model in which the individual channels are modeled by a Markov chain [18]. It was confirmed that the present white noise approximation is acceptable for such realistic models.

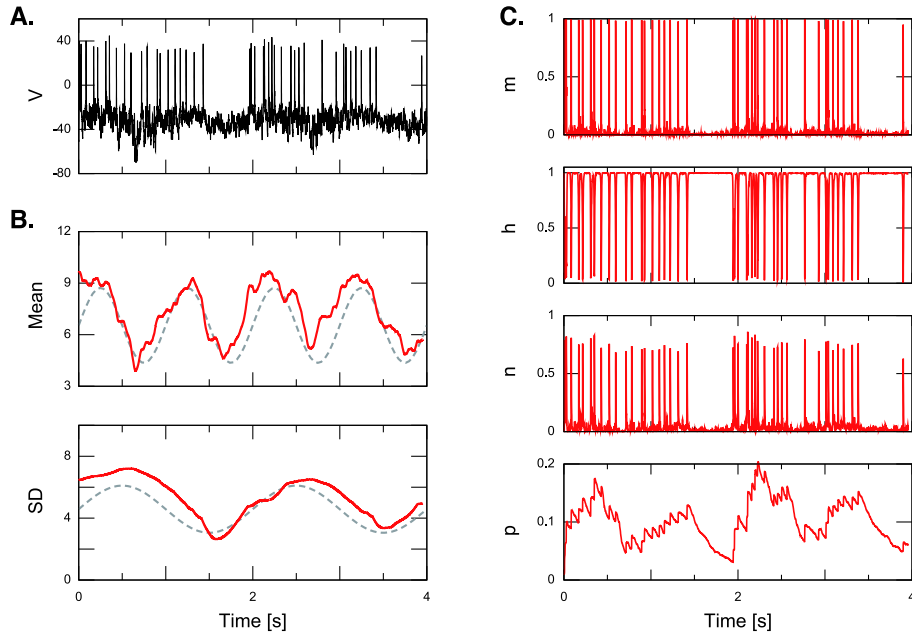

Figure 4: Estimation of input signals and ion channel states from experimental data. A. Voltage trace recorded intracellularly *in vitro*. Fluctuating current, sinusoidally modulated mean and standard deviation (SD), was injected to the neuron. B. Estimation of the time-varying mean and SD of the input. The gray dashed and red lines represent the true and the estimates, respectively. C. Estimation of the ion channels state. The red lines represent the estimates of the gating variables.

Fourth, we ignored the possible nonlinear effects in dendritic conduction such as dendritic spike and backpropagating action potential. It would be worthwhile to consider augmenting the model by dividing into multiple compartments as has been done in Huys et al. [6].

Fifth, in analyzing experimental data, we employed fixed functions for the ionic currents and the rate constants and assumed that some of the intrinsic parameters are known. It may be possible to infer the maximal ionic conductances using the particle filter method developed by Huys and Paninski [23], but their method is not able to identify the ionic currents and the rate constants. In our examination of biological data, we have explored parameters empirically from current-voltage data. It would be an important direction of this study to develop the method such that models are selected solely from the voltage trace.

**Acknowledgments**

This study was supported by Support Center for Advanced Telecommunications Technology Research, Foundation; Yazaki Memorial Foundation for Science and Technology; and Ritsumeikan University Research Funding Research Promoting Program "Young Scientists (Start-up)", "General Research" to R.K., Grant-in-Aid for Young Scientists (B) from the MEXT Japan (22700323) to Y.T., Grants-in-Aid for Scientific Research from the MEXT Japan (20300083, 23115510) to S.S., and the Center for Neurosciences LC554, Grant No. AV0Z50110509 and the Grant Agency of the Czech Republic, project P103/11/0282 to P.L.

## Footnotes

\*Webpage: http://www.ritsumei.ac.jp/∼r-koba84/index.html

# References

[1] Koch, C. (1999) *Biophysics of Computation: Information Processing in Single Neurons.* Oxford University Press.

[2] Lansky, P. (1983) *Math. Biosci.* **67**: 247-260.

[3] Lansky, P. & Ditlevsen S. (2008) *Biol. Cybern.* **99**: 253-262.

[4] Rudolph, M., Piwkowska, Z., Badoual, M., Bal, T. & Destexhe, A. (2004) *J. Neurophysiol.* **91**: 2884-2896.

[5] Pospischil, M., Piwkowska, Z., Bal, T. & Destexhe, A. (2009) *Neurosci.* **158**: 545-552.

[6] Huys, Q.J.M., Ahrens, M.B. & Paninski, L. (2006) *J. Neurophysiol.* **96**: 872-890.

[7] Shinomoto, S., Sakai, S. & Funahashi, S. (1999) *Neural Comput.* **11**: 935-951.

[8] DeWeese, M.R. & Zador, A.M. (2006) *J. Neurosci.* **26**: 12206-12218.

[9] Fox, R.F. (1997) *Biophys. J.* **72**: 2068-2074.

[10] Burkitt, A.N. (2006) *Biol. Cybern.* **95**: 1-19.

[11] Kitagawa, G. & Gersh, W. (1996) *Smoothness priors analysis of time series.* New York: Springer-Verlag.

[12] Pospischil, M., Toledo-Rodriguez, M., Monier, C., Piwkowska, Z., Bal, T., Fregnac, Y., Markram, H. & Destexhe, A. (2008) *Biol. Cybern.* **99**: 427-441.

[13] Dempster, A.P., Laird, N.M. & Rubin, D.B. (1977) *J. R. Stat. Soc.* **39**: 1-38.

[14] Smith, A.C. & Brown, E.N. (2003) *Neural Comput.* **15**: 965-991.

[15] Eden, U.T., Frank, L.M., Barbieri, R., Solo, V. & Brown, E.N., (2004) *Neural Comput.* **16**: 971-998.

[16] Paninski, L., Ahmadian, Y., Ferreira, D.G., Koyama, S., Rad, K.R., Vidne, M., Vogelstein, J. & Wu, W. (2010) *J. Comput. Neurosci.* **29**: 107-126.

[17] Koyama, S., Pérez-Bolde, L.C., Shalizi, C.R. & Kass, R.E. (2010) *J. Am. Stat. Assoc.* **105**: 170-180.

[18] Schneidman, E., Freedman, B. & Segev, I. (1998) *Neural Comput.* **10**: 1679-1703.

[19] Tsubo, Y., Takada, M., Reyes, A. D. & Fukai, T. (2007) *Eur. J. Neurosci.* **25**: 3429-3441.

[20] Kobayashi, R., Tsubo, Y. & Shinomoto, S. (2009) *Front. Comput. Neurosci.* **3**: 9.

[21] Lansky, P., Sanda, P. & He, J. (2006) *J. Comput. Neurosci.* **21**: 211-223.

[22] Kobayashi, R., Shinomoto, S. & Lansky, P. (2011) *Neural Comput.* **23**: 3070-3093.

[23] Huys, Q.J.M. & Paninski, L. (2009) *PLoS Comput. Biol.* **5**: e1000379.

